# Small-Variance Asymptotics for Exponential Family Dirichlet Process Mixture Models

**Ke Jiang, Brian Kulis**
Department of CSE
The Ohio State University
{jiangk,kulis}@cse.ohio-state.edu

**Michael I. Jordan**
Departments of EECS and Statistics
University of California at Berkeley
jordan@cs.berkeley.edu

## Abstract

Sampling and variational inference techniques are two standard methods for inference in probabilistic models, but for many problems, neither approach scales effectively to large-scale data. An alternative is to relax the probabilistic model into a non-probabilistic formulation which has a scalable associated algorithm. This can often be fulfilled by performing small-variance asymptotics, i.e., letting the variance of particular distributions in the model go to zero. For instance, in the context of clustering, such an approach yields connections between the $k$-means and EM algorithms. In this paper, we explore small-variance asymptotics for exponential family Dirichlet process (DP) and hierarchical Dirichlet process (HDP) mixture models. Utilizing connections between exponential family distributions and Bregman divergences, we derive novel clustering algorithms from the asymptotic limit of the DP and HDP mixtures that features the scalability of existing hard clustering methods as well as the flexibility of Bayesian nonparametric models. We focus on special cases of our analysis for discrete-data problems, including topic modeling, and we demonstrate the utility of our results by applying variants of our algorithms to problems arising in vision and document analysis.

## 1 Introduction

An enduring challenge for machine learning is in the development of algorithms that scale to truly large data sets. While probabilistic approaches—particularly Bayesian models—are flexible from a modeling perspective, lack of scalable inference methods can limit applicability on some data. For example, in clustering, algorithms such as $k$-means are often preferred in large-scale settings over probabilistic approaches such as Gaussian mixtures or Dirichlet process (DP) mixtures, as the $k$-means algorithm is easy to implement and scales to large data sets.

In some cases, links between probabilistic and non-probabilistic models can be made by applying asymptotics to the variance (or covariance) of distributions within the model. For instance, connections between probabilistic and standard PCA can be made by letting the covariance of the data likelihood in probabilistic PCA tend toward zero [1, 2]; similarly, the $k$-means algorithm may be obtained as a limit of the EM algorithm when the covariances of the Gaussians corresponding to each cluster goes to zero. Besides providing a conceptual link between seemingly quite different approaches, small-variance asymptotics can yield useful alternatives to probabilistic models when the data size becomes large, as the non-probabilistic models often exhibit more favorable scaling properties. The use of such techniques to derive scalable algorithms from rich probabilistic models is still emerging, but provides a promising approach to developing scalable learning algorithms.

This paper explores such small-variance asymptotics for clustering, focusing on the DP mixture. Existing work has considered asymptotics over the Gaussian DP mixture [3], leading to $k$-means-like algorithms that do not fix the number of clusters upfront. This approach, while an important first step, raises the question of whether we can perform similar asymptotics over distributions other

than the Gaussian. We answer in the affirmative by showing how such asymptotics may be applied to the exponential family distributions for DP mixtures; such analysis opens the door to a new class of scalable clustering algorithms and utilizes connections between Bregman divergences and exponential families. We further extend our approach to hierarchical nonparametric models (specifically, the hierarchical Dirichlet process (HDP) [4]), and we view a major contribution of our analysis to be the development of a general hard clustering algorithm for grouped data.

One of the primary advantages of generalizing beyond the Gaussian case is that it opens the door to novel scalable algorithms for discrete-data problems. For instance, visual bag-of-words [5] have become a standard representation for images in a variety of computer vision tasks, but many existing probabilistic models in vision cannot scale to the size of data sets now commonly available. Similarly, text document analysis models (e.g., LDA [6]) are almost exclusively discrete-data problems. Our analysis covers such problems; for instance, a particular special case of our analysis is a hard version of HDP topic modeling. We demonstrate the utility of our methods by exploring applications in text and vision.

**Related Work:** In the non-Bayesian setting, asymptotics for the expectation-maximization algorithm for exponential family distributions were studied in [7]. The authors showed a connection between EM and a general $k$-means-like algorithm, where the squared Euclidean distance is replaced by the Bregman divergence corresponding to exponential family distribution of interest. Our results may be viewed as generalizing this approach to the Bayesian nonparametric setting. As discussed above, our results may also be viewed as generalizing the approach of [3], where the asymptotics were performed for the DP mixture with a Gaussian likelihood, leading to a $k$-means-like algorithm where the number of clusters is not fixed upfront. Note that our setting is considerably more involved than either of these previous works, particularly since we will require an appropriate technique for computing an asymptotic marginal likelihood. Other connections between hard clustering and probabilistic models were explored in [8], which proposes a "Bayesian $k$-means" algorithm by performing a maximization-expectation algorithm.

## 2   Background

In this section, we briefly review exponential family distributions, Bregman divergences, and the Dirichlet process mixture model.

### 2.1   The Exponential Family

Consider the exponential family with natural parameter $\boldsymbol{\theta} = \{\theta_j\}_{j=1}^d \in \mathbb{R}^d$; then the exponential family probability density function can be written as [9]:

$$p(\boldsymbol{x} \,|\, \boldsymbol{\theta}) = \exp\big(\langle \boldsymbol{x}, \boldsymbol{\theta} \rangle - \psi(\boldsymbol{\theta}) - h(\boldsymbol{x})\big),$$

where $\psi(\boldsymbol{\theta}) = \log \int \exp(\langle \boldsymbol{x}, \boldsymbol{\theta} \rangle - h(\boldsymbol{x})) d\boldsymbol{x}$ is the log-partition function. Here we assume for simplicity that $\boldsymbol{x}$ is a minimal sufficient statistic for the natural parameter $\boldsymbol{\theta}$. $\psi(\boldsymbol{\theta})$ can be utilized to compute the mean and covariance of $p(\boldsymbol{x} \,|\, \boldsymbol{\theta})$; in particular, the expected value is given by $\nabla \psi(\boldsymbol{\theta})$, and the covariance is $\nabla^2 \psi(\boldsymbol{\theta})$.

**Conjugate Priors**: In a Bayesian setting, we will require a prior distribution over the natural parameter $\boldsymbol{\theta}$. A convenient property of the exponential family is that a conjugate prior distribution of $\boldsymbol{\theta}$ exists; in particular, given any specific distribution in the exponential family, the conjugate prior can be parametrized as [11]:

$$p(\boldsymbol{\theta} \,|\, \tau, \eta) = \exp\big(\langle \boldsymbol{\theta}, \tau \rangle - \eta \psi(\boldsymbol{\theta}) - m(\tau, \eta)\big).$$

Here, the $\psi(\cdot)$ function is the same as that of the likelihood function. Given a data point $\boldsymbol{x}_i$, the posterior distribution of $\boldsymbol{\theta}$ has the same form as the prior, with $\tau \to \tau + \boldsymbol{x}_i$ and $\eta \to \eta + 1$.

**Relationship to Bregman Divergences**: Let $\phi : S \to \mathbb{R}$ be a differentiable, strictly convex function defined on a convex set $S \subseteq \mathbb{R}^d$. The Bregman divergence for any pair of points $\boldsymbol{x}, \boldsymbol{y} \in S$ is defined as $D_\phi(\boldsymbol{x}, \boldsymbol{y}) = \phi(\boldsymbol{x}) - \phi(\boldsymbol{y}) - \langle \boldsymbol{x} - \boldsymbol{y}, \nabla \phi(\boldsymbol{y}) \rangle$, and can be viewed as a generalized distortion measure. An important result connecting Bregman divergences and exponential families was discussed in [7] (see also [10, 11]), where a bijection between the two was established. A key consequence of this result is that we can equivalently parameterize both $p(\boldsymbol{x} \,|\, \boldsymbol{\theta})$ and $p(\boldsymbol{\theta} \,|\, \tau, \eta)$ in terms of the

expectation $\boldsymbol{\mu}$:

$$p(\boldsymbol{x} \,|\, \boldsymbol{\theta}) = p(\boldsymbol{x} \,|\, \boldsymbol{\mu}) = \exp(-D_\phi(\boldsymbol{x}, \boldsymbol{\mu})) f_\phi(\boldsymbol{x}),$$

$$p(\boldsymbol{\theta} \,|\, \tau, \eta) = p(\boldsymbol{\mu} \,|\, \tau, \eta) = \exp\left(-\eta D_\phi\left(\frac{\tau}{\eta}, \boldsymbol{\mu}\right)\right) g_\phi(\tau, \eta),$$

where $\phi(\cdot)$ is the Legendre-conjugate function of $\psi(\cdot)$ (denoted as $\phi = \psi^*$), $f_\phi(\boldsymbol{x}) = \exp(\phi(\boldsymbol{x}) - h(\boldsymbol{x}))$, and $\boldsymbol{\mu}$ is the expectation parameter which satisfies $\boldsymbol{\mu} = \nabla\psi(\boldsymbol{\theta})$ (and also $\boldsymbol{\mu} = \boldsymbol{\theta}^*$). The Bregman divergence representation provides a natural way to parametrize the exponential family distributions with its expectation parameter and, as we will see, we will find it convenient to work with this form.

## 2.2 Dirichlet Process Mixture Models

The Dirichlet Process (DP) mixture model is a Bayesian nonparametric mixture model [12]; unlike most parametric mixture models (Bayesian or otherwise), the number of clusters in a DP mixture is not fixed upfront. Using the exponential family parameterized by the expectation $\boldsymbol{\mu}_c$, the likelihood for a data point can be expressed as the following *infinite* mixture:

$$p(\boldsymbol{x}) = \sum_{c=1}^{\infty} \pi_c p(\boldsymbol{x} \,|\, \boldsymbol{\mu}_c) = \sum_{c=1}^{\infty} \pi_c \exp(-D_\phi(\boldsymbol{x}, \boldsymbol{\mu}_c)) f_\phi(\boldsymbol{x}).$$

Even though there are conceptually an infinite number of clusters, the nonparametric prior over the mixing weights causes the weights $\pi_c$ to decay exponentially. Moreover, a simple collapsed Gibbs sampler can be employed for performing inference in this model [13]; this Gibbs sampler will form the basis of our asymptotic analysis. Given a data set $\{\boldsymbol{x}_1, ..., \boldsymbol{x}_n\}$, the state of the Markov chain is the set of cluster indicators $\{z_1, ..., z_n\}$ as well as the cluster means of the currently-occupied clusters (the mixing weights have been integrated out). The Gibbs updates for $z_i$, $(i = 1, \ldots, n)$, are given by the following conditional probabilities:

$$P(z_i = c \,|\, z_{-i}, \boldsymbol{x}_i, \boldsymbol{\mu}) = \frac{n_{-i,c}}{\mathbf{Z}(n-1+\alpha)} p(\boldsymbol{x}_i \,|\, \boldsymbol{\mu}_c)$$

$$P(z_i = c_{new} \,|\, z_{-i}, \boldsymbol{x}_i, \boldsymbol{\mu}) = \frac{\alpha}{\mathbf{Z}(n-1+\alpha)} \int p(\boldsymbol{x}_i \,|\, \boldsymbol{\mu}) dG_0,$$

where $\mathbf{Z}$ is the normalizing constant, $n_{-i,c}$ is the number of data points (excluding $\boldsymbol{x}_i$) that are currently assigned to cluster $c$, $G_0$ is a prior over $\boldsymbol{\mu}$, and $\alpha$ is the *concentration parameter* that determines how likely we are to start a new cluster. If we choose to start a new cluster during the Gibbs update, we sample its mean from the posterior distribution obtained from the prior distribution $G_0$ and the single observation $\boldsymbol{x}_i$. After performing Gibbs moves on the cluster indicators, we update the cluster means $\boldsymbol{\mu}_c$ by sampling from the posterior of $\boldsymbol{\mu}_c$ given the data points assigned to cluster $c$.

## 3 Hard Clustering for Exponential Family DP Mixtures

Our goal is to analyze what happens as we perform small-variance asymptotics on the exponential family DP mixture when running the collapsed Gibbs sampler described earlier, and we begin by considering how to scale the covariance in an exponential family distribution. Given an exponential family distribution $p(\boldsymbol{x} \,|\, \boldsymbol{\theta})$ with natural parameter $\boldsymbol{\theta}$ and log-partition function $\psi(\boldsymbol{\theta})$, consider a *scaled* exponential family distribution whose natural parameter is $\tilde{\boldsymbol{\theta}} = \beta\boldsymbol{\theta}$ and log-partition function is $\tilde{\psi}(\tilde{\boldsymbol{\theta}}) = \beta\psi(\tilde{\boldsymbol{\theta}}/\beta)$, where $\beta > 0$. The following result characterizes the relationship between the mean and covariance of the original and scaled exponential family distributions.

**Lemma 3.1.** *Denote $\boldsymbol{\mu}(\boldsymbol{\theta})$ as the mean, and $cov(\boldsymbol{\theta})$ as the covariance, of $p(\boldsymbol{x} \,|\, \boldsymbol{\theta})$ with log-partition $\psi(\boldsymbol{\theta})$. Given a scaled exponential family with $\tilde{\boldsymbol{\theta}} = \beta\boldsymbol{\theta}$ and $\tilde{\psi}(\tilde{\boldsymbol{\theta}}) = \beta\psi(\tilde{\boldsymbol{\theta}}/\beta)$, the mean $\tilde{\boldsymbol{\mu}}(\tilde{\boldsymbol{\theta}})$ of the scaled distribution is $\boldsymbol{\mu}(\boldsymbol{\theta})$ and the covariance, $\tilde{cov}(\tilde{\boldsymbol{\theta}})$, is $cov(\boldsymbol{\theta})/\beta$.*

This lemma follows directly from $\tilde{\boldsymbol{\mu}}(\tilde{\boldsymbol{\theta}}) = \nabla_{\tilde{\theta}}\tilde{\psi}(\tilde{\boldsymbol{\theta}}) = \beta\nabla_{\tilde{\theta}}\psi(\tilde{\boldsymbol{\theta}}/\beta) = \nabla_{\theta}\psi(\tilde{\boldsymbol{\theta}}/\beta) = \nabla_{\theta}\psi(\boldsymbol{\theta}) = \boldsymbol{\mu}(\boldsymbol{\theta})$, and $\tilde{cov}(\tilde{\boldsymbol{\theta}}) = \nabla_{\tilde{\theta}}^2(\tilde{\psi}(\tilde{\boldsymbol{\theta}})) = \beta\nabla_{\tilde{\theta}}(\nabla_{\tilde{\theta}}\psi(\tilde{\boldsymbol{\theta}}/\beta)) = \frac{1}{\beta} \times \nabla_{\theta}^2\psi(\tilde{\boldsymbol{\theta}}/\beta) = \frac{1}{\beta} \times \nabla_{\theta}^2\psi(\boldsymbol{\theta}) = cov(\boldsymbol{\theta})/\beta$. It is perhaps intuitively simpler to observe what happens to the distribution using the

Bregman divergence representation. Recall that the generating function $\phi$ for the Bregman divergence is given by the Legendre-conjugate of $\psi$. Using standard properties of convex conjugates, we see that the conjugate of $\tilde{\psi}$ is simply $\tilde{\phi} = \beta\phi$. The Bregman divergence representation for the scaled distribution is given by

$$p(\boldsymbol{x} \mid \tilde{\boldsymbol{\theta}}) = p(\boldsymbol{x} \mid \tilde{\boldsymbol{\mu}}) = \exp(-D_{\tilde{\phi}}(\boldsymbol{x}, \tilde{\boldsymbol{\mu}}))f_{\tilde{\phi}}(\boldsymbol{x}) = \exp(-\beta D_{\phi}(\boldsymbol{x}, \boldsymbol{\mu}))f_{\beta\phi}(\boldsymbol{x}),$$

where the last equality follows from Lemma 3.1 and the fact that, for a Bregman divergence, $D_{\beta\phi}(\cdot, \cdot) = \beta D_{\phi}(\cdot, \cdot)$. Thus, as $\beta$ increases under the above scaling, the mean is fixed while the distribution becomes increasingly concentrated around the mean.

Next we consider the prior distribution under the scaled exponential family. When scaling by $\beta$, we also need to scale the hyper-parameters $\tau$ and $\eta$, namely $\tau \to \tau/\beta$ and $\eta \to \eta/\beta$. This gives the following prior written using the Bregman divergence, where we are now explicitly conditioning on $\beta$:

$$p(\tilde{\boldsymbol{\theta}} \mid \tau, \eta, \beta) = \exp\left(-\frac{\eta}{\beta}D_{\tilde{\phi}}\left(\frac{\tau/\beta}{\eta/\beta}, \boldsymbol{\mu}\right)\right)g_{\tilde{\phi}}\left(\frac{\tau}{\beta}, \frac{\eta}{\beta}\right) = \exp\left(-\eta D_{\phi}\left(\frac{\tau}{\eta}, \boldsymbol{\mu}\right)\right)g_{\tilde{\phi}}\left(\frac{\tau}{\beta}, \frac{\eta}{\beta}\right).$$

Finally, we compute the marginal likelihood for $\boldsymbol{x}$ by integrating out $\tilde{\boldsymbol{\theta}}$, as it will be necessary for the Gibbs sampler. Standard algebraic manipulations yield the following:

$$
\begin{aligned}
p(\boldsymbol{x} \mid \tau, \eta, \beta) &= \int p(\boldsymbol{x} \mid \tilde{\boldsymbol{\theta}}) \times p(\tilde{\boldsymbol{\theta}} \mid \tau, \eta, \beta)d\tilde{\boldsymbol{\theta}} \\
&= f_{\tilde{\phi}}(\boldsymbol{x}) \cdot g_{\tilde{\phi}}\left(\frac{\tau}{\beta}, \frac{\eta}{\beta}\right)A_{(\tilde{\phi}, \tau, \eta, \beta)}(\boldsymbol{x})\int \exp\left(-(\beta+\eta)D_{\phi}\left(\frac{\beta\boldsymbol{x}+\tau}{\beta+\eta}, \tilde{\boldsymbol{\mu}}(\tilde{\boldsymbol{\theta}})\right)\right)d\tilde{\boldsymbol{\theta}} \\
&= f_{\tilde{\phi}}(\boldsymbol{x}) \cdot g_{\tilde{\phi}}\left(\frac{\tau}{\beta}, \frac{\eta}{\beta}\right)A_{(\tilde{\phi}, \tau, \eta, \beta)}(\boldsymbol{x}) \cdot \beta^d \cdot \int \exp\left(-(\beta+\eta)D_{\phi}\left(\frac{\beta\boldsymbol{x}+\tau}{\beta+\eta}, \boldsymbol{\mu}(\boldsymbol{\theta})\right)\right)d\boldsymbol{\theta}.
\end{aligned}
\tag{1}
$$

Here, $A_{(\tilde{\phi}, \tau, \eta, \beta)}(\boldsymbol{x}) = \exp\left(-(\beta\phi(\boldsymbol{x}) + \eta\phi(\frac{\tau}{\eta}) - (\beta+\eta)\phi(\frac{\beta\boldsymbol{x}+\tau}{\beta+\eta}))\right)$, which arises when combining the Bregman divergences from the likelihood and the prior.

Now we make the following key insight, which will allow us to perform the necessary asymptotics. We can write the integral from the last line above (denoted $I$ below) via Laplace's method. Since $D_{\phi}(\frac{\beta\boldsymbol{x}+\tau}{\beta+\eta}, \boldsymbol{\mu})$ has a local minimum (which is global in this case) at $\hat{\boldsymbol{\theta}} = \hat{\boldsymbol{\mu}}^* = (\frac{\beta\boldsymbol{x}+\tau}{\beta+\eta})^*$, we have:

$$
\begin{aligned}
I &= \exp\left(-(\beta+\eta)D_{\phi}\left(\frac{\beta\boldsymbol{x}+\tau}{\beta+\eta}, \hat{\boldsymbol{\mu}}\right)\right)\left(\frac{2\pi}{\beta+\eta}\right)^{d/2}\left|\frac{\partial^2 D_{\phi}(\frac{\beta\boldsymbol{x}+\tau}{\beta+\eta}, \hat{\boldsymbol{\mu}})}{\partial\boldsymbol{\theta}\partial\boldsymbol{\theta}^T}\right|^{-1/2} + O\left(\frac{1}{\beta}\right) \\
&= \left(\frac{2\pi}{\beta+\eta}\right)^{d/2}\left|\frac{\partial^2 D_{\phi}(\frac{\beta\boldsymbol{x}+\tau}{\beta+\eta}, \hat{\boldsymbol{\mu}})}{\partial\boldsymbol{\theta}\partial\boldsymbol{\theta}^T}\right|^{-1/2} + O\left(\frac{1}{\beta}\right)
\end{aligned}
\tag{2}
$$

where $\frac{\partial^2 D_{\phi}(\frac{\beta\boldsymbol{x}+\tau}{\beta+\eta}, \hat{\boldsymbol{\mu}})}{\partial\boldsymbol{\theta}\partial\boldsymbol{\theta}^T} = \text{cov}(\hat{\boldsymbol{\theta}})$ is the covariance matrix of the likelihood function instantiated at $\hat{\boldsymbol{\theta}}$ and approaches $\text{cov}(\boldsymbol{x}^*)$ when $\beta$ goes to $\infty$. Note that the exponential term equals one since the divergence inside is 0.

## 3.1 Asymptotic Behavior of the Gibbs Sampler

We now have the tools to consider the Gibbs sampler for the exponential family DP mixture as we let $\beta \to \infty$. As we will see, we will obtain a general $k$-means-like hard clustering algorithm which utilizes the appropriate Bregman divergence in place of the squared Euclidean distance, and also can vary the number of clusters. Recall the conditional probabilities for performing Gibbs moves on the cluster indicators $z_i$, where we now are considering the scaled distributions:

$$
\begin{aligned}
P(z_i = c \mid z_{-i}, \boldsymbol{x}_i, \beta, \boldsymbol{\mu}) &= \frac{n_{-i,c}}{\mathbf{Z}}\exp(-\beta D_{\phi}(\boldsymbol{x}_i, \boldsymbol{\mu_c}))f_{\tilde{\phi}}(\boldsymbol{x}_i) \\
P(z_i = c_{new} \mid z_{-i}, \boldsymbol{x}_i, \beta, \boldsymbol{\mu}) &= \frac{\alpha}{\mathbf{Z}}p(\boldsymbol{x}_i \mid \tau, \eta, \beta),
\end{aligned}
$$

where $\mathbf{Z}$ is a normalization factor, and the marginal probability $p(\boldsymbol{x}_i \mid \tau, \eta, \beta)$ is given by the derivations in (1) and (2). Now, we consider the asymptotic behavior of these probabilities as $\beta \to \infty$. We

note that

$$\lim_{\beta \to \infty} \frac{\beta \boldsymbol{x}_i + \tau}{\beta + \eta} = \boldsymbol{x}_i, \quad \text{and} \quad \lim_{\beta \to \infty} A_{(\tilde{\phi}, \tau, \eta, \beta)}(\boldsymbol{x}_i) = \exp(-\eta(\phi(\tau/\eta) - \phi(\boldsymbol{x}_i))),$$

and that the Laplace approximation error term goes to zero as $\beta \to \infty$. Further, we define $\alpha$ as a function of $\beta$, $\eta$, and $\tau$ (but independent of the data):

$$\alpha = \left( g_{\tilde{\phi}}\left(\frac{\tau}{\beta}, \frac{\eta}{\beta}\right) \cdot \left(\frac{2\pi}{\beta + \eta}\right)^{d/2} \cdot \beta^d \right)^{-1} \cdot \exp(-\beta\lambda),$$

for some $\lambda$. After canceling out the $f_{\tilde{\phi}}(\boldsymbol{x}_i)$ terms from all probabilities, we can then write the Gibbs probabilities as

$$P(z_i = c \,|\, z_{-i}, \boldsymbol{x}_i, \beta, \boldsymbol{\mu}) = \frac{n_{-i,c} \cdot \exp(-\beta D_\phi(\boldsymbol{x}_i, \boldsymbol{\mu}_c))}{C_{\boldsymbol{x}_i} \cdot \exp(-\beta\lambda) + \sum_{j=1}^{k} n_{-i,j} \cdot \exp(-\beta D_\phi(\boldsymbol{x}_i, \boldsymbol{\mu}_j))}$$

$$P(z_i = c_{new} \,|\, z_{-i}, \boldsymbol{x}_i, \beta, \boldsymbol{\mu}) = \frac{C_{\boldsymbol{x}_i} \cdot \exp(-\beta\lambda)}{C_{\boldsymbol{x}_i} \cdot \exp(-\beta\lambda) + \sum_{j=1}^{k} n_{-i,j} \cdot \exp(-\beta D_\phi(\boldsymbol{x}_i, \boldsymbol{\mu}_j))},$$

where $C_{\boldsymbol{x}_i}$ approaches a positive, finite constant for a given $\boldsymbol{x}_i$ as $\beta \to \infty$. Now, all of the above probabilities will become binary as $\beta \to \infty$. More specifically, all the $k+1$ values will be increasingly dominated by the smallest value of $\{D_\phi(\boldsymbol{x}_i, \boldsymbol{\mu}_1), \ldots, D_\phi(\boldsymbol{x}_i, \boldsymbol{\mu}_k), \lambda\}$. As $\beta \to \infty$, only the smallest of these values will receive a non-zero probability. That is, the data point $\boldsymbol{x}_i$ will be assigned to the *nearest* cluster with a *divergence* at most $\lambda$. If the closest mean has a divergence greater than $\lambda$, we start a new cluster containing only $\boldsymbol{x}_i$.

Next, we show that sampling $\boldsymbol{\mu}_c$ from the posterior distribution is achieved by simply computing the empirical mean of a cluster in the limit. During Gibbs sampling, once we have performed one complete set of Gibbs moves on the cluster assignments, we need to sample the $\boldsymbol{\mu}_c$ conditioned on all assignments and observations. If we let $n_c$ be the number of points assigned to cluster $c$, then the posterior distribution (parameterized by the expectation parameter) for cluster $c$ is

$$p(\boldsymbol{\mu}_c \,|\, X, \boldsymbol{z}, \tau, \eta, \beta) \propto p(X_c \,|\, \boldsymbol{\mu}_c, \beta) \times p(\boldsymbol{\mu}_c \,|\, \tau, \eta, \beta) \propto \exp\left(-(\beta n_c + \eta) D_\phi\left(\frac{\sum_{i=1}^{n_c} \beta \boldsymbol{x}_i^c + \tau}{\beta n_c + \eta}, \boldsymbol{\mu}\right)\right),$$

where $X$ is all the data, $X_c = \{\boldsymbol{x}_1^c, ..., \boldsymbol{x}_{n_c}^c\}$ is the set of points currently assigned to cluster $c$, and $\boldsymbol{z}$ is the set of all current assignments. We can see that the mass of the posterior distribution becomes concentrated around the sample mean $\frac{\sum_{i=1}^{n_c} \boldsymbol{x}_i}{n_c}$ as $\beta \to \infty$. In other words, after we determine the assignments of data points to clusters, we update the means as the sample mean of the data points in each cluster. This is equivalent to the standard $k$-means cluster mean update step.

## 3.2 Objective function and algorithm

From the above asymptotic analysis of the Gibbs sampler, we observe a new algorithm which can be utilized for hard clustering. It is as simple as the popular $k$-means algorithm, but also provides the ability to adapt the number of clusters depending on the data as well as incorporate different distortion measures. The algorithm description is as follows:

- Initialization: input data $\boldsymbol{x}_1, \ldots, \boldsymbol{x}_n$, $\lambda > 0$, and $\mu_1 = \frac{1}{n} \sum_{i=1}^{n} \boldsymbol{x}_n$
- Assignment: for each data point $\boldsymbol{x}_i$, compute the Bregman divergence $D_\phi(\boldsymbol{x}_i, \boldsymbol{\mu}_c)$ to all existing clusters. If $\min_c D_\phi(\boldsymbol{x}_i, \boldsymbol{\mu}_c) \leq \lambda$, then $z_{i,c_0} = 1$ where $c_0 = \mathrm{argmin}_c D_\phi(\boldsymbol{x}_i, \boldsymbol{\mu}_c)$; otherwise, start a new cluster and set $z_{i,c_{new}} = 1$;
- Mean Update: compute the cluster mean for each cluster, $\mu_j = \frac{1}{|l_j|} \sum_{\boldsymbol{x} \in l_j} \boldsymbol{x}$, where $l_j$ is the set of points in the $j$-th cluster.

We iterate between the assignment and mean update steps until local convergence. Note that the initialization used here—placing all data points into a single cluster—is not necessary, but is one natural way to initialize the algorithm. Also note that the algorithm depends heavily on the choice of $\lambda$; heuristics for selecting $\lambda$ were briefly discussed for the Gaussian case in [3], and we will follow this approach (generalized in the obvious way to Bregman divergences) for our experiments.

We can easily show that the underlying objective function for our algorithm is quite similar to that in [3], replacing the squared Euclidean distance with an appropriate Bregman divergence. Recall that the squared Euclidean distance is the Bregman divergence corresponding to the Gaussian distribution. Thus, the objective function in [3] can be seen as a special case of our work. The objective function optimized by our derived algorithm is the following:

$$\min_{\{l_c\}_{c=1}^k} \quad \sum_{c=1}^{k} \sum_{\boldsymbol{x} \in l_c} D_{\phi}(\boldsymbol{x}, \boldsymbol{\mu}_c) + \lambda k \tag{3}$$

where $k$ is the total number of clusters, $\phi$ is the conjugate function of the log-partition function of the chosen exponential family distribution, and $\boldsymbol{\mu}_c$ is the sample mean of cluster $c$. The penalty term $\lambda$ controls the tradeoff between the likelihood and the model complexity, where a large $\lambda$ favors small model complexity (i.e., fewer clusters) while a small $\lambda$ favors more clusters. Given the above objective function, our algorithm can be shown to monotonically decrease the objective function value until convergence to some local minima. We omit the proof here as it is almost identical as the proof for Theorem 3.1 in [3].

## 4 Extension to Hierarchies

A key benefit of the Bayesian approach is its natural ability to form hierarchical models. In the context of clustering, a hierarchical mixture allows one to cluster multiple groups of data—each group is clustered into a set of local clusters, but these local clusters are shared among the groups (i.e., sets of local clusters across groups form global clusters, with a shared global mean). For Bayesian nonparametric mixture models, one way of achieving such hierarchies arises via the hierarchical Dirichlet Process (HDP) [4], which provides a nonparametric approach to allow sharing of clusters among a set of DP mixtures.

In this section, we will briefly sketch out the extension of our analysis to the HDP mixture, which yields a natural extension of our methods to groups of data. Given space considerations, and the fact that the resulting algorithm turns out to reduce to Algorithm 2 from [3] with the squared Euclidean distance replaced by an appropriate Bregman divergence, we will omit the full specification of the algorithm here. However, despite the similarity to the existing Gaussian case, we do view the extension to hierarchies as a promising application of our analysis. In particular, our approach opens the door to hard hierarchical algorithms over discrete data, such as text, and we briefly discuss an application of our derived algorithm to topic modeling.

We assume that there are $J$ data sets (groups) which we index by $j = 1, ..., J$. Data point $\boldsymbol{x}_{ij}$ refers to data point $i$ from set $j$. The HDP model can be viewed as clustering each data set into local clusters, but where each local cluster is associated to a global mean. Global means may be shared across data sets. When performing the asymptotics, we require variables for the global means $(\boldsymbol{\mu}_1, ..., \boldsymbol{\mu}_g)$, the associations of data points to local clusters, $z_{ij}$, and the associations of local clusters to global means, $v_{jt}$, where $t$ indexes the local clusters for a data set. A standard Gibbs sampler considers updates on all of these variables, and in the nonparametric setting does not fix the number of local or global clusters.

The tools from the previous section may be nearly directly applied to the hierarchical case. As opposed to the flat model, the hard HDP requires two parameters: a value $\lambda_{top}$ which is utilized when starting a global (top-level) cluster, and a value $\lambda_{bottom}$ which is utilized when starting a local cluster. The resulting hard clustering algorithm first performs local assignment moves on the $z_{ij}$, then updates the local cluster assignments, and finally updates all global means.

The resulting objective function that is monotonically minimized by our algorithm is given as follows:

$$\min_{\{l_c\}_{c=1}^k} \quad \sum_{c=1}^{k} \sum_{\boldsymbol{x}_{ij} \in l_c} D_{\phi}(\boldsymbol{x}_{ij}, \boldsymbol{\mu}_c) + \lambda_{bottom} t + \lambda_{top} k, \tag{4}$$

where $k$ is the total number of global clusters and $t$ is the total number of local clusters. The bottom-level penalty term $\lambda_{bottom}$ controls both the number of local and top-level clusters, where larger $\lambda_{bottom}$ tends to give fewer local clusters and more top-level clusters. Meanwhile, the top-level penalty term $\lambda_{top}$, as in the one-level case, controls the tradeoff between the likelihood and model complexity.

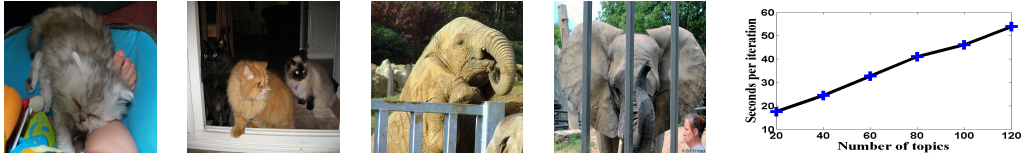

Figure 1: (Left) Example images from the ImageNet data (Persian cat and elephant categories). Each image is represented via a discrete visual-bag-of-words histogram. Clustering via an asymptotic multinomial DP mixture considerably outperforms the asymptotic Gaussian DP mixture; see text for details. (Right) Elapsed time per iteration in seconds of our topic modeling algorithm when running on the NIPS data, as a function of the number of topics.

## 5 Experiments

We conclude with a brief set of experiments highlighting applications of our analysis to discrete-data problems, namely image clustering and topic modeling. For all experiments, we randomly permute the data points at each iteration, as this tends to improve results (as discussed previously, unlike standard $k$-means, the order in which the data points are processed impacts the resulting clusters).

**Image Clustering.** We first explore an application of our techniques to image clustering, focusing on the ImageNet data [14]. We utilize a subset of this data for quantitative experiments, sampling 100 images from 10 different categories of this data set (Persian cat, African elephant, fire engine, motor scooter, wheelchair, park bench, cello, French horn, television, and goblet), for a total of 1000 images. Each image is processed via standard visual-bag-of-words: SIFT is densely applied on top of image patches in image, and the resulting SIFT vectors are quantized into 1000 visual words. We use the resulting histograms as our discrete representation for an image, as is standard. Some example images from this data set are shown in Figure 1.

We explore whether the discrete version of our hard clustering algorithm based on a multinomial DP mixture outperforms the Gaussian mixture version (i.e., DP-means); this will validate our generalization beyond the Gaussian setting. For both the Gaussian and multinomial cases, we utilize a farthest-first approach for both selecting $\lambda$ as well as initializing the clusters (see [3] for a discussion of farthest-first for selecting $\lambda$).

We compute the normalized mutual information (NMI) between the true clusters and the results of the two algorithms on this difficult data set. The Gaussian version performs poorly, achieving an NMI of .06 on this data, whereas the hard multinomial version achieves a score of .27. While the multinomial version is far from perfect, it performs significantly better than DP-means. Scalability to large data sets is clearly feasible, given that the method scales linearly in the number of data points. Note that comparisons between the Gibbs sampler and the corresponding hard clustering algorithm for the Gaussian case were considered in [3], where experiments on several data sets showed comparable clustering accuracy results between the sampler and the hard clustering method. Furthermore, for a fully Bayesian model that places a prior on the concentration parameter, the sampler was shown to be considerably slower than the corresponding hard clustering method. Given the similarity of the sampler for the Gaussian and multinomial case, we expect similar behavior with the multinomial Gibbs sampler.

**Illustration: Scalable Hard Topic Models.** We also highlight an application to topic modeling, by providing some qualitative results over two common document collections. Utilizing our general algorithm for a hard version of the multinomial HDP is straightforward. In order to apply the hard hierarchical algorithm to topic modeling, we simply utilize the discrete KL-divergence in the hard exponential family HDP, since topic modeling for text uses a multinomial distribution for the data likelihood.

To test topic modeling using our asymptotic approach, we performed analyses using the NIPS 1-12[1] and the NYTimes [15] datasets. For the NIPS dataset, we use the whole dataset, which contains 1740 total documents, 13649 words in the vocabulary, and 2,301,375 total words. For the NYTimes

|   | NIPS | NYTimes |
|---|------|---------|
| 1 | neurons, memory, patterns, activity, response, neuron, stimulus, firing, cortex, recurrent, pattern, spike, stimuli, delay, responses | team, game, season, play, games, point, player, coach, win, won, guy, played, playing, record, final |
| 2 | neural, networks, state, weight, states, results, synaptic, threshold, large, time, systems, activation, small, work, weights | percent, campaign, money, fund, quarter, federal, public, pay, cost, according, income, half, term, program, increase |
| 3 | training, hidden, recognition, layer, performance, probability, parameter, error, speech, class, weights, trained, algorithm, approach, order | president, power, government, country, peace, trial, public, reform, patriot, economic, past, clear, interview, religious, early |
| 4 | cells, visual, cell, orientation, cortical, connection, receptive, field, center, tuning, low, ocular, present, dominance, fields | family, father, room, line, shares, recount, told, mother, friend, speech, expression, won, offer, card, real |
| 5 | energy, solution, methods, function, solutions, local, equations, minimum, hopfield, temperature, adaptation, term, optimization, computational, procedure | company, companies, stock, market, business, billion, firm, computer, analyst, industry, internet, chief, technology, customer, number |
| 6 | noise, classifier, classifiers, note, margin, noisy, regularization, generalization, hypothesis, multiclasses, prior, cases, boosting, fig, pattern | right, human, decision, need, leadership, foundation, number, question, country, strike, set, called, support, law, train |

Table 1: Sample topics inferred from the NIPS and NYTimes datasets by our hard multinomial HDP algorithm.

dataset, we randomly sampled 2971 documents with 10171 vocabulary words, and 853,451 words in total; we also eliminated low-frequency words (those with less than ten occurrences). The prevailing metric to measure the goodness of topic models is perplexity; however, this is based on the predictive probability, which has no counterpart in the hard clustering case. Furthermore, ground truth for topic models is difficult to obtain. This makes quantitative comparisons difficult for topic modeling, and so we therefore focus on qualitative results. Some sample topics (with the corresponding top 15 terms) discovered by our approach from both the NIPS and NYTimes datasets are given in Table 1; we can see that the topics appear to be quite reasonable. Also, we highlight the scalability of our approach: the number of iterations needed for convergence on these data sets ranges from 13 to 25, and each iteration completes in under one minute (see the right side of Figure 1). In contrast, for sampling methods, it is notoriously difficult to detect convergence, and generally a large number of iterations is required. Thus, we expect this approach to scale favorably to large data sets.

## 6 Conclusion

We considered a general small-variance asymptotic analysis for the exponential family DP and HDP mixture model. Crucially, this analysis allows us to move beyond the Gaussian distribution in such models, and opens the door to new clustering applications, such as those involving discrete data. Our analysis utilizes connections between Bregman divergences and exponential families, and results in a simple and scalable hard clustering algorithm which may be viewed as generalizing existing non-Bayesian Bregman clustering algorithms [7] as well as the DP-means algorithm [3]. Due to the prevalence of discrete data in modern computer vision and information retrieval, we hope our algorithms will find use for a variety of large-scale data analysis tasks. We plan to continue to focus on the difficult problem of quantitative evaluations comparing probabilistic and non-probabilistic methods for clustering, particularly for topic models. We also plan to compare our algorithms with recent online inference schemes for topic modeling, particularly the online LDA [16] and online HDP [17] algorithms.

**Acknowledgements.** This work was supported by NSF award IIS-1217433 and by the ONR under grant number N00014-11-1-0688.

## Footnotes

[1]http://www.cs.nyu.edu/ roweis/data.html

# References

[1] M. E. Tipping and C. M. Bishop. Probabilistic principal component analysis. *Journal of the Royal Statistical Society, Series B*, 21(3):611–622, 1999.

[2] S. Roweis. EM algorithms for PCA and SPCA. In *Advances in Neural Information Processing Systems*, 1998.

[3] B. Kulis and M. I. Jordan. Revisiting k-means: New algorithms via Bayesian nonparametrics. In *Proceedings of the 29th International Conference on Machine Learning*, 2012.

[4] Y. W. Teh, M. I. Jordan, M. J. Beal, and D. M. Blei. Hierarchical Dirichlet processes. *Journal of the American Statistical Association*, 101(476):1566–1581, 2006.

[5] L. Fei-Fei and P. Perona. A Bayesian hierarchical model for learning natural scene categories. In *IEEE Conference on Computer Vision and Patterns Recognition*, 2005.

[6] D. Blei, A. Ng, and M. I. Jordan. Latent Dirichlet allocation. *Journal of Machine Learning Research*, 3:993–1022, 2003.

[7] A. Banerjee, S. Merugu, I. S. Dhillon, and J. Ghosh. Clustering with Bregman divergences. *Journal of Machine Learning Research*, 6:1705–1749, 2005.

[8] K. Kurihara and M. Welling. Bayesian k-means as a "Maximization-Expectation" algorithm. *Neural Computation*, 21(4):1145–1172, 2008.

[9] O. Barndorff-Nielsen. *Information and Exponential Families in Statistical Theory*. Wiley Publishers, 1978.

[10] J. Forster and M. K. Warmuth. Relative expected instantaneous loss bounds. In *Proceedings of 13th Conference on Computational Learning Theory*, 2000.

[11] A. Agarwal and H. Daume. A geometric view of conjugate priors. *Machine Learning*, 81(1):99–113, 2010.

[12] N. Hjort, C. Holmes, P. Mueller, and S. Walker. *Bayesian Nonparametrics: Principles and Practice*. Cambridge University Press, Cambridge, UK, 2010.

[13] R. M. Neal. Markov chain sampling methods for Dirichlet process mixture models. *Journal of Computational and Graphical Statistics*, 9:249–265, 2000.

[14] J. Deng, W. Dong, R. Socher, L.-J. Li, K. Li, and L. Fei-Fei. ImageNet: A large-scale hierarchical image database. In *IEEE Conference on Computer Vision and Patterns Recognition*, 2009.

[15] A. Frank and A. Asuncion. UCI Machine Learning Repository, 2010.

[16] M. D. Hoffman, D. M. Blei, and F. Bach. Online learning for Latent Dirichlet Allocation. In *Advances in Neural Information Processing Systems*, 2010.

[17] C. Wang, J. Paisley, and D. M. Blei. Online variational inference for the hierarchical Dirichlet process. In *Proceedings of the 14th International Conference on Artificial Intelligence and Statistics*, 2011.

